# Dynamics of Learning in Recurrent Feature-Discovery Networks

**Todd K. Leen**
Department of Computer Science and Engineering
Oregon Graduate Institute of Science & Technology
Beaverton, OR 97006-1999

## Abstract

The self-organization of recurrent feature-discovery networks is studied from the perspective of dynamical systems. Bifurcation theory reveals parameter regimes in which multiple equilibria or limit cycles coexist with the equilibrium at which the networks perform principal component analysis.

## 1 Introduction

Oja (1982) made the remarkable observation that a simple model neuron with an Hebbian adaptation rule develops into a filter for the first principal component of the input distribution. Several researchers have extended Oja's work, developing networks that perform a complete principal component analysis (PCA). Sanger (1989) proposed an algorithm that uses a single layer of weights with a set of cascaded feedback projections to force nodes to filter for the principal components. This architecture singles out a particular node for each principal component. Oja (1989) and Oja and Karhunen (1985) give a related algorithm that projects inputs onto an orthogonal basis spanning the principal subspace, but does not necessarily filter for the principal components themselves.

In another class of models, nodes are forced to learn different statistical features by a set of lateral connections. Rubner and Schulten (1990) use cascaded lateral connections; the $i^{th}$ node receives signals from the input and all nodes $j$ with $j < i$. The lateral connections are modified by an anti-Hebbian learning rule that tends to de-correlate the node responses. Like Sanger's scheme, this architecture singles out a particular node for each principal component. Kung and Diamantaras (1990) propose a different learning rule on the same network topology. Foldiak (1989) simulates a network with full lateral connectivity, but does not discuss convergence.

The goal of this paper is to help form a more complete picture of feature-discovery models that use lateral signal flow. We discuss two models with particular emphasis on their learning dynamics. The models incorporate Hebbian and anti-Hebbian adaptation, and recurrent lateral connections. We give stability analyses and derive bifurcation diagrams for the models. Stability analysis gives a lower bound on the rate of adaptation the lateral connections, below which the equilibrium corresponding to PCA is unstable. Bifurcation theory provides a description of the behavior near loss of stability. The bifurcation analyses reveal stable equilibria in which the weight vectors from the input are combinations of the eigenvectors of the input correlation. Limit cycles are also found.

## 2    The Single-Neuron Model

In Oja's model the input, $x \in R^N$, is a random vector assumed to be drawn from a stationary probability distribution. The vector of synaptic weights is denoted $\omega$ and the post-synaptic response is linear; $y = x \cdot \omega$. The continuous-time, ensemble averaged form of the learning rule is

$$
\begin{aligned}
\dot{\omega} &= \; <xy> - <y^2> \omega \\
&= \; R\omega - (\omega \cdot R\omega)\,\omega
\end{aligned}
\tag{1}
$$

where $< \ldots >$ denotes the average over the ensemble of inputs, and $R = <x\,x^T>$ is the correlation matrix. The unit-magnitude eigenvectors of $R$ are denoted $e_i, \; i = 1 \ldots N$ and are assumed to be ordered in decreasing magnitude of the associated eigenvalues $\lambda_1 > \lambda_2 > \ldots > \lambda_N > 0$. Oja shows that the weight vector asymptotically approaches $\pm e_1$. The variance of the node's response is thus maximized and the node acts as a filter for the first principal component of the input distribution.

## 3    Extending the Single Neuron Model

To extend the model to a system of $M \leq N$ nodes we consider a set of linear neurons with weight vectors (called the forward weights) $\omega_1 \ldots \omega_M$ connecting each to the $N-$dimensional input. Without interactions between the nodes in the array, all $M$ weight vectors would converge to $\pm e_1$.

We consider two approaches to building interactions that force nodes to filter for different statistical features. In the first approach an internode potential is constructed. This formulation results in a *non-local* model. The model is made local by introducing lateral connections that naturally acquire *anti*-Hebbian adaptation. For reasons that will become clear, the resulting model is referred to as a minimal coupling scheme. In the second approach, we write equations of motion of the forward weights based directly on (1). The evolution of the lateral connection strengths will follow a simple anti-Hebbian rule.

### 3.1    Minimal Coupling

The response of the $i^{th}$ node in the array is taken to be linear in the input

$$
y_i = x \cdot \omega_i.
\tag{2}
$$

The adaptation of the forward weights is derived from the potential

$$
\begin{aligned}
U &= -\frac{1}{2} \sum_j^M <y_j^2> + \frac{C}{2} \sum_{j,k;j \neq k}^M <y_j\, y_k>^2 \\
&= -\frac{1}{2} \sum_j^M (\omega_j \cdot R\omega_j) + \frac{C}{2} \sum_{j,k;j \neq k}^M (\omega_j \cdot R\omega_k)^2,
\end{aligned} \tag{3}
$$

where $C$ is a coupling constant. The first term of $U$ generates the Hebb law, while the second term penalizes correlated node activity (Yuille *et al.* 1989). The equations of motion are constructed to perform gradient descent on $U$ with a term added to bound the weight vectors,

$$
\begin{aligned}
\dot{\omega}_i &= -\nabla_{\omega_i} U - <y_i^2> \omega_i \\
&= <x\, y_i> - C \sum_{j \neq i}^M <y_i\, y_j> <x\, y_j> - <y_i^2> \omega_i \\
&= R\omega_i - C \sum_{j \neq i}^M (\omega_i \cdot R\omega_j)\, R\omega_j - (\omega_i \cdot R\omega_i)\, \omega_i.
\end{aligned} \tag{4}
$$

Note that $\omega_i$ refers to the weight vector from the input to the $i^{th}$ node, *not* the $i^{th}$ component of the weight vector.

Equation (4) is *non-local* as it involves correlations, $<y_i\, y_j>$, between nodes. In order to provide a purely local adaptation, we introduce a symmetric matrix of lateral connections

$$
\begin{aligned}
\eta_{ij} \quad & i,j = 1, \ldots, M \\
\eta_{ii} &= 0.
\end{aligned}
$$

These evolve according to

$$
\begin{aligned}
\dot{\eta}_{ij} &= -d\,(\eta_{ij} + C <y_i\, y_j>) \\
&= -d\,(\eta_{ij} + C\, \omega_i \cdot R\omega_j)
\end{aligned} \tag{5}
$$

where $d$ is a rate constant. In the limit of fast adaptation (large $d$)

$$
\eta_{ij} \to -C <y_i\, y_j>.
$$

With this limiting behavior in mind, we replace (4) with

$$
\begin{aligned}
\dot{\omega}_i &= <xy_i> + \sum_{j \neq i}^M \eta_{ij} <xy_j> - <y_i^2> \omega_i \\
&= R\omega_i + \sum_{j \neq i}^M \eta_{ij}\, R\omega_j - (\omega_i \cdot R\omega_i)\, \omega_i.
\end{aligned} \tag{6}
$$

Equations (5) and (6) specify the adaptation of the network.

Notice that the response of the $i^{th}$ node is given by (2) and is thus independent of the signals carried on the lateral connections. In this sense the lateral signals affect node plasticity but not node response. This minimal coupling can also be derived as a low-order approximation to the model in §3.2 below.

### 3.1.1   Stability and Bifurcation

By inspection the weight dynamics given by (5) and (6) have an equilibrium at

$$X_0 \equiv (\omega_i = e_i, \ \eta_{ij} = 0). \tag{7}$$

At this equilibrium the outputs are the first $M$ principal components of input vectors. In suitable coordinates the linear part of the equations of motion break into block diagonal form with any possible instabilities constrained to $3 \times 3$ sub-blocks. Details of the stability and bifurcation analysis are given in Leen (1991). The principal component subspace is *always* asymptotically stable. However the equilibrium $X_0$ is linearly stable if and only if

$$d > d_0 = \frac{(\lambda_i - \lambda_j)^2 (\lambda_i + \lambda_j)}{\lambda_i^2 + \lambda_j^2} \tag{8}$$

$$C > C_0 = \frac{1}{\lambda_i + \lambda_j}, \quad 1 \le (i,j) \le M. \tag{9}$$

At $C_0$ or $d_0$ there is a qualitative change (a bifurcation) in the learning dynamics. If the condition on $d$ is violated, then there is a Hopf bifurcation to oscillating weights. At the critical value $C_0$ there is a bifurcation to multiple equilibria. The bifurcation normal form was found by Liapunov-Schmidt reduction (see e.g. Golubitsky and Schaeffer 1984) performed at the bifurcation point $(X_0, C_0)$. To deal effectively with the large dimensional phase space of the network, the calculations were performed on a symbolic algebra program.

At the critical point $(X_0, C_0)$ there is a supercritical pitchfork bifurcation. Two *unstable* equilibria appear near $X_0$ for $C > C_0$. At these equilibria the forward weights are mixtures of $e_M$ and $e_{M-1}$ and the lateral connection strengths are non-zero. Generically one expects a saddle-node bifurcation. However $X_0$ is an equilibrium for all values of $C$, and the system has an inversion symmetry. These conditions preclude the saddle-node and transcritical bifurcations, and we are left with the pitchfork.

The position of *stable* equilibria away from $(X_0, C_0)$ can be found by examining terms of order five and higher in the bifurcation expansion. Alternatively we examine the bifurcation from the homogeneous solution, $X_h$, in which all weight vectors are proportional to $e_1$. For a system of two nodes this equilibrium is asymptotically stable provided

$$C < C_h \equiv \min \left\{ \begin{array}{c} (\lambda_1 - \lambda_2)/(2\lambda_1\lambda_2) \\ 1/\lambda_1 \end{array} \right\} \tag{10}$$

If $\lambda_1 < 3\lambda_2$, then there is a supercritical pitchfork bifurcation at $C_h$. Two *stable* equilibria emerge from $X_h$ for $C > C_h$. At these stable equilibria, the forward weight vectors are mixtures of the first two correlation eigenvectors and the lateral connection strengths are nonzero.

The complete bifurcation diagram for a system of two nodes is shown in Fig. 1. The upper portion of the figure shows the bifurcation at $(X_0, C_0)$. The horizontal line corresponds to the PCA equilibrium $X_0$. This equilibrium is stable (heavy line) for

$C > C_0$, and unstable (light line) for $C < C_0$. The subsidiary, unstable, equilibria that emerge from $(X_0, C_0)$ lie on the light, parabolic branches of the top diagram. Calculations indicate that the form of this bifurcation is independent of the number of nodes, and of the input dimension. Of course the value of $C_0$ increases with increasing number of nodes, c.f. (9).

The lower portion of Fig. 1 shows the bifurcation from $(X_h, C_h)$ for a system of two nodes. The horizontal line corresponds to the homogeneous equilibrium $X_h$. This is stable for $C < C_h$ and unstable for $C > C_h$. The stable equilibria consisting of mixtures of the correlation eigenvectors lie on the heavy parabolic branches of the diagram. For networks with more nodes, there are presumably further bifurcations along the supercritical stable branches emerging from $(X_h, C_h)$; equilibria with qualitatively different eigenvector mixtures are observed in simulations.

Each inset in the figure shows equilibrium forward weight vectors for both nodes in a two-node network. These configurations were generated by numerical integration of the equations of motion (5) and (6). The correlation matrix corresponds to an ensemble of noise vectors with short-range correlations between the components. Simulations of the corresponding *discrete*, pattern-by-pattern learning rule confirm the form of the weight vectors shown here.

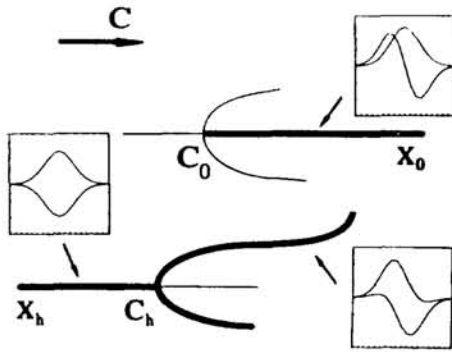

Figure 1: Bifurcation diagram for the minimal model

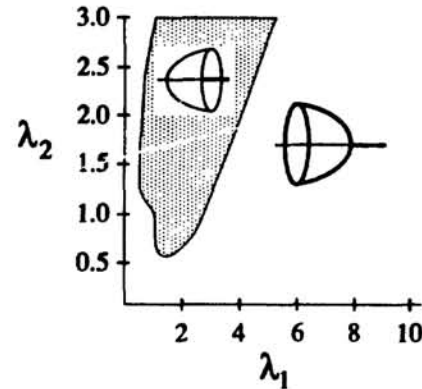

Fig 2: Regions in the $(\lambda_1, \lambda_2)$ plane corresponding to supercritical (shaded) and subcritical (unshaded) Hopf bifurcation.

### 3.2  Full Coupling

In a more conventional coupling scheme, the signals carried on the lateral connections affect the node activities directly. For linear node response, the vector of activities is given by

$$y = (1 - \eta)^{-1} \omega x \equiv u \omega x \qquad (11)$$

where $y \in R^M$, $\eta$ is the $M \times M$ matrix of lateral connection strengths and $\omega$ is an $M \times N$ matrix whose $i^{th}$ row is the forward weight vector to the $i^{th}$ node. The adaptation rule is

$$\dot{\omega} = <yx^T> - Diag(<yy^T>)\omega \qquad (12)$$

$$\dot{\eta} = D\eta - C<yy^T>, \quad \eta_{ii} = 0, \qquad (13)$$

where $D$ and $C$ are constants and $Diag$ sets the off-diagonal elements of its argument equal to zero. This system also has the PCA equilibrium $X_0$. This is linearly stable if

$$D > 0 \tag{14}$$

$$C > C_0 \equiv \frac{D}{\lambda_i + \lambda_j} + \frac{(\lambda_i - \lambda_j)^2}{\lambda_i^2 + \lambda_j^2}. \tag{15}$$

Equation (14) tells us that the PCA equilibrium is structurally unstable without the $D\eta$ term in (13). Without this term, the model reduces to that given by Foldiak (1989). That the latter generally does not converge to the PCA equilibrium is consistent with the condition in (14).

If, on the other hand, the condition on $C$ is violated then the network undergoes a Hopf bifurcation leading to oscillations. Depending on the eigenvalue spectrum of the input correlation, this bifurcation may be subcritical (with stable limit cycles near $X_0$ for $C < C_0$), or supercritical (with unstable limit cycles near $X_0$ for $C > C_0$). Figure 2 shows the corresponding regions in the $(\lambda_1, \lambda_2)$ plane for a network of two nodes with $D = 1$. Simulations show that even in the supercritical regime, stable limit cycles are found for $C < C_0$, and for $C > C_0$ sufficiently close to $C_0$. This suggests that the complete bifurcation diagram in the super-critical regime is shaped like the bottom of a wine bottle, with only the indentation shown in figure 2. Under the approximation $u \approx 1 + \eta$, the super-critical regime is significantly narrowed.

## 4  Discussion

The primary goal of this study has been to give a theoretical description of learning in feature-discovery models; in particular models that use lateral interactions to ensure that nodes tune to different statistical features. The models presented here have several different limit sets (equilibria and cycles) whose stability and location in the weight space depends on the relative learning rates in the network, and on the eigenvalue spectrum of the input correlation. We have applied tools from bifurcation theory to qualitatively describe the location and determine stability of these different limiting solutions. This theoretical approach provides a unifying framework within which similar algorithms can be studied.

Both models have equilibria at which the network performs PCA. In addition, the minimal model has stable equilibria for which the forward weight vectors are mix-tures of the correlation eigenvectors. Both models have regimes in which the weight vectors oscillate. The model given by Rubner $et$ $al.$ (1990) also loses stability through Hopf bifurcation for small values of the lateral learning rate.

The minimal values of $C$ in (9) and (15) for the stability of the PCA equilibrium can become quite large for small correlation eigenvalues. These stringent conditions can be ameliorated in both models by the replacement

$$d\,\eta_{ij} \rightarrow (<y_i^2> + <y_j^2>)\,\eta_{ij}.$$

However in the minimal model, this leads to degenerate bifurcations which have not been thoroughly examined.

Finally, it remains to be seen whether the techniques employed here extend to similar systems with non-linear node activation (e.g. Carlson 1991) or to the problem of locating multiple minima in cost functions for *supervised* learning models.

## Acknowledgments

This work was supported by the Office of Naval Research under contract N00014-90-1349 and by DARPA grant MDA 972-88-J-1004 to the Department of Computer Science and Engineering. The author thanks Bill Baird for stimulating e-mail discussion.

## References

Carlson, A. (1991) Anti-Hebbian learning in a non-linear neural network *Biol. Cybern.*, 64:171–176.

Foldiak, P. (1989) Adaptive network for optimal linear feature extraction. In *Proceedings of the IJCNN*, pages I 401–405.

Golubitsky, Martin and Schaeffer, David (1984) *Singularities and Groups in Bifurcation Theory, Vol. I.* Springer-Verlag, New York.

Kung, S. and Diamantaras K. (1990) A neural network learning algorithm for adaptive principal component extraction (APEX). In *Proceedings of the IEEE International Conference on Acoustics Speech and Signal Processing*, pages 861–864.

Leen, T. K. (1991) Dynamics of learning in linear feature-discovery networks. *Network : Computation in Neural Systems*, to appear.

Oja, E. (1982) A simplified neuron model as a principal component analyzer. *J. Math. Biology*, 15:267–273.

Oja, E. (1989) Neural networks, principal components, and subspaces. *International Journal of Neural Systems*, 1:61–68.

Oja, E. and Karhunen, J. (1985) On stochastic approximation of the eigenvectors and eigenvalues of the expectation of a random matrix. *J. of Math. Anal. and Appl.*, 106:69–84.

Rubner, J. and Schulten K. (1990) Development of feature detectors by self-organization: A network model. *Biol. Cybern.*, 62:193–199.

Sanger, T. (1989) An optimality principle for unsupervised learning. In D.S. Touretzky, editor, *Advances in Neural Information Processing Systems 1*. Morgan Kauffmann.

Yuille, A.L, Kammen, D.M. and Cohen, D.S. (1989) Quadrature and the development of orientation selective cortical cells by Hebb rules. *Biol. Cybern.*, 61:183–194.